# Deterministic Annealing Variant of the EM Algorithm

**Naonori Ueda**     **Ryohei Nakano**
ueda@cslab.kecl.ntt.jp     nakano@cslab.kecl.ntt.jp
NTT Communication Science Laboratories
Hikaridai, Seika-cho, Soraku-gun,
Kyoto 619-02 Japan

## Abstract

We present a deterministic annealing variant of the EM algorithm for maximum likelihood parameter estimation problems. In our approach, the EM process is reformulated as the problem of minimizing the thermodynamic free energy by using the *principle of maximum entropy* and *statistical mechanics analogy*. Unlike simulated annealing approaches, this minimization is *deterministically* performed. Moreover, the derived algorithm, unlike the conventional EM algorithm, can obtain better estimates free of the initial parameter values.

## 1   INTRODUCTION

The Expectation-Maximization (EM) algorithm (Dempster, Laird & Rubin, 1977) is an iterative statistical technique for computing maximum likelihood parameter estimates from incomplete data. It has generally been employed to a wide variety of parameter estimation problems. Recently, the EM algorithm has also been successfully employed as the learning algorithm of the hierarchical mixture of experts (Jordan & Jacobs, 1993). In addition, it has been found to have some relationship to the learning of the Boltzmann machines (Byrne, 1992).

This algorithm has attractive features such as reliable global convergence, low cost per iteration, economy of storage, and ease of programming, but it is not free from problems in practice. The serious practical problem associated with the algorithm

is the local maxima problem. This problem makes the performance dependent on the initial parameter value. Indeed, the EM algorithm should be performed from as wide a choice of starting values as possible according to some ad hoc criterion.

To overcome this problem, we adopt the principle of statistical mechanics. Namely, by using the principle of maximum entropy, the thermodynamic free energy is defined as an effective cost function that depends on the *temperature*. The maximization of *log-likelihood* is done by minimizing the cost function. Unlike simulated annealing (Geman & Geman, 1984) where stochastic search is performed on the given energy surface, this cost function is *deterministically* optimized at each temperature.

Such deterministic annealing (DA) approach has been successfully adopted for vector quantization or clustering problems (Rose *et al.*, 1992; Buhmann *et al.*, 1993; Wong, 1993). Recently, Yuile *et al.*(Yuile, Stolorz, & Utans, 1994) have shown that the EM algorithm can be used in conjunction with the DA. In our previous paper, independent of Yuile's work, we presented a new EM algorithm with DA for mixture density estimation problems (Ueda & Nakano, 1994). The aim of this paper is to generalize our earlier work and to derive a DA variant of the general EM algorithm. Since the EM algorithm can be used not only for the mixture estimation problems but also for other parameter estimation problems, this generalization is expected to be of value in practice.

## 2   GENERAL THEORY OF THE EM ALGORITHM

Suppose that a measure space $\mathcal{Y}$ of "unobservable data" exists corresponding to a measure space $\mathcal{X}$ of "observable data". An observable data sample $x(\in \mathcal{X})$ with density $p(x; \Theta)$ is called incomplete and $(x, y)$ with joint density $p(x, y; \Theta)$ is called complete, where $y$ is an unobservable data sample [1] corresponding to $x$. Note that $x \in \mathcal{R}^n$ and $y \in \mathcal{R}^m$. $\Theta$ is parameter of the density distribution to be estimated.

Given incomplete data samples $X = \{x_k | k = 1, \cdots, N\}$, the goal of the EM algorithm is to compute the maximum-likelihood estimate of $\Theta$ that maximizes the following log-likelihood function:

$$L(\Theta; X) = \sum_{k=1}^{N} \log p(x_k; \Theta), \tag{1}$$

by using the following complete data log-likelihood function:

$$L_c(\Theta; X) = \sum_{k=1}^{N} \log p(x_k, y_k; \Theta). \tag{2}$$

In the EM algorithm, the parameter $\Theta$ is iteratively estimated. Suppose that $\Theta^{(t)}$ denotes the current estimate of $\Theta$ after the $t$th iteration of the algorithm. Then $\Theta^{(t+1)}$ at the next iteration is determined by the following two steps:

**E-step**: Compute the $Q$-function defined by the conditional expectation of the complete data log-likelihood given $X$ and $\Theta^{(t)}$:

$$Q(\Theta; \Theta^{(t)}) \overset{\text{def}}{=} E\{L_c(\Theta; X)|X, \Theta^{(t)}\}. \tag{3}$$

**M-step**: Set $\Theta^{(t+1)}$ equal to $\Theta$ which maximizes $Q(\Theta, \Theta^{(t)})$.

It has theoretically been shown that an iterative procedure for maximizing $Q$ over $\Theta$ will cause the likelihood $L$ to monotonically increase, e.g., $L(\Theta^{(t+1)}) \geq L(\Theta^{(t)})$. Eventually, $L(\Theta^{(t)})$ converges to a local maximum. The EM algorithm is especially useful when the maximization of the $Q$-function can be more easily performed than that of $L$.

By substituting Eq. 2 into Eq. 3, we have

$$Q(\Theta; \Theta^{(t)}) = \sum_{k=1}^{N} \int \cdots \int \{\log p(x_k, y_k; \Theta)\} \prod_{j=1}^{N} p(y_j|x_j; \Theta^{(t)}) dy_1 \cdots dy_N$$

$$= \sum_{k=1}^{N} \int \{\log p(x_k, y_k; \Theta)\} p(y_k|x_k; \Theta^{(t)}) dy_k. \tag{4}$$

$\Theta$ that maximizes $Q(\Theta; \Theta^{(t)})$ should satisfy $\partial Q/\partial\Theta = 0$, or equivalently,

$$\sum_{k=1}^{N} \int \{\frac{\partial}{\partial\Theta} \log p(x_k, y_k; \Theta)\} p(y_k|x_k; \Theta^{(t)}) dy_k = 0. \tag{5}$$

Here, $p(y_k|x_k, \Theta^{(t)})$ denotes the posterior and can be computed by the following Bayes rule:

$$p(y_k|x_k; \Theta^{(t)}) = \frac{p(x_k, y_k; \Theta^{(t)})}{\int p(x_k, y_k; \Theta^{(t)}) dy_k}. \tag{6}$$

It can be interpreted that the missing information is estimated by using the posterior. However, because the reliability of the posterior highly depends on the parameter $\Theta^{(t)}$, the performance of the EM algorithm is sensitive to an initial parameter value $\Theta^{(0)}$. This has often caused the algorithm to become trapped by some local maxima. In the next section, we will derive a new variant of the EM algorithm as an attempt at global maximization of the $Q$-function in the EM process.

## 3 DETERMINISTIC ANNEALING APPROACH

### 3.1 DERIVATION OF PARAMETERIZED POSTERIOR

Instead of the posterior given in Eq. 6, we introduce another posterior $f(y_k|x_k)$. The function form of $f$ will be specified later. Using $f(y_k|x_k)$, we consider a new function instead of $Q$, say $E$, defined as:

$$E \overset{\text{def}}{=} \sum_{k=1}^{N} \int \{-\log p(x_k, y_k; \Theta)\} f(y_k|x_k) dy_k. \tag{7}$$

(Note: $E$ is always nonnegative.) One can easily see that $(-E)$ is also the conditional expectation of the complete data log-likelihood but it differs from $Q$ in that the expectation is taken with respect to $f(\boldsymbol{y}_k|\boldsymbol{x}_k)$ instead of the posterior given by Eq. 6. In other words, if $f(\boldsymbol{y}_k|\boldsymbol{x}_k) = p(\boldsymbol{y}_k|\boldsymbol{x}_k;\boldsymbol{\Theta}^{(t)})$, then $E \equiv -Q$.

Since we do not have *a priori* knowledge about $f(\boldsymbol{y}_k|\boldsymbol{x}_k)$, we apply the *principle of maximum entropy* to specify it. That is, by maximizing the entropy given by:

$$H = -\sum_{k=1}^{N}\int \{\log f(\boldsymbol{y}_k|\boldsymbol{x}_k)\}f(\boldsymbol{y}_k|\boldsymbol{x}_k)d\boldsymbol{y}_k, \tag{8}$$

with respect to $f$, under the constraints of Eq. 7 and $\int f d\boldsymbol{y}_k = 1$, we can obtain the following Gibbs distribution:

$$f(\boldsymbol{y}_k|\boldsymbol{x}_k) = \frac{1}{Z_{\boldsymbol{x}_k}}\exp\{-\beta(-\log p(\boldsymbol{x}_k,\boldsymbol{y}_k;\boldsymbol{\Theta}))\}, \tag{9}$$

where $Z_{\boldsymbol{x}_k} = \int \exp\{-\beta(-\log p(\boldsymbol{x}_k,\boldsymbol{y}_k;\boldsymbol{\Theta}))\}d\boldsymbol{y}_k$, and is called the *partition function*. The parameter $\beta$ is the Lagrange multiplier determined by the value $E$. From an analogy of the *annealing*, $1/\beta$ corresponds to the *"temperature"*.

By simplifying Eq. 9, we obtain a new posterior parameterized by $\beta$,

$$f(\boldsymbol{y}_k|\boldsymbol{x}_k) = \frac{p(\boldsymbol{x}_k,\boldsymbol{y}_k;\boldsymbol{\Theta})^\beta}{\int p(\boldsymbol{x}_k,\boldsymbol{y}_k;\boldsymbol{\Theta})^\beta d\boldsymbol{y}_k}. \tag{10}$$

Clearly, when $\beta = 1$, $f(\boldsymbol{y}_k|\boldsymbol{x}_k)$ reduces to the original posterior given in Eq. 6. The effect of $\beta$ will be explained later.

Since $\boldsymbol{x}_1,\ldots,\boldsymbol{x}_N$ are identically and independently distributed observations, the partition function $Z_\beta(\boldsymbol{\Theta})$ for $\boldsymbol{X}$ becomes $\prod_k Z_{\boldsymbol{x}_k}$. Therefore,

$$Z_\beta(\boldsymbol{\Theta}) = \prod_{k=1}^{N}\int p(\boldsymbol{y}_k,\boldsymbol{y}_k;\boldsymbol{\Theta})^\beta d\boldsymbol{y}_k. \tag{11}$$

Once the partition function is obtained explicitly, using statistical mechanics analogy, we can define the *free energy* as an effective cost function that depends on the temperature:

$$F_\beta(\boldsymbol{\Theta}) \stackrel{\text{def}}{=} -\frac{1}{\beta}\log Z_\beta(\boldsymbol{\Theta})$$

$$= -\frac{1}{\beta}\sum_{k=1}^{N}\log\int p(\boldsymbol{x}_k,\boldsymbol{y}_k;\boldsymbol{\Theta})^\beta d\boldsymbol{y}_k. \tag{12}$$

At equilibrium, it is well known that a thermodynamic system settles into a configuration that minimizes its free energy. Hence, $\boldsymbol{\Theta}$ should satisfy $\partial F_\beta(\boldsymbol{\Theta})/\partial\boldsymbol{\Theta} = 0$. It follows that

$$\sum_{k=1}^{N}\int \{\frac{\partial}{\partial\boldsymbol{\Theta}}\log p(\boldsymbol{x}_k,\boldsymbol{y}_k;\boldsymbol{\Theta})\}f(\boldsymbol{y}_k|\boldsymbol{x}_k)d\boldsymbol{y}_k = 0. \tag{13}$$

Interestingly, we have arrived at the same equation as the result of the maximization of the $Q$-function, except that the posterior $p(\boldsymbol{y}_k|\boldsymbol{x}_k;\boldsymbol{\Theta}^{(t)})$ in Eq. 5 is replaced by $f(\boldsymbol{y}_k|\boldsymbol{x}_k)$.

## 3.2 ANNEALING VARIANT OF THE EM ALGORITHM

Let $Q_\beta(\Theta; \Theta^{(t)})$ be the expectation of the complete data log-likelihood by the parameterized posterior $f(y_k|x_k)$. Then, the following deterministic annealing variant of the EM algorithm can be naturally derived to maximize $-F_\beta(\Theta)$.

**[Annealing EM (AEM) algorithm]**
1. Set $\beta \leftarrow \beta_{min}(0 < \beta_{min} \ll 1)$.
2. *Arbitrarily* choose an initial estimate $\Theta^{(0)}$. Set $t \leftarrow 0$.
3. Iterate the following two steps until convergence[2]:
   E-step: Compute

$$Q_\beta(\Theta; \Theta^{(t)}) = \sum_{k=1}^{N} \int \{\log p(x_k, y_k; \Theta)\} \frac{p(x_k, y_k; \Theta^{(t)})^\beta}{\int p(x_k, y_k; \Theta^{(t)})^\beta dy_k} dy_k. \quad (14)$$

   M-step: Set $\Theta^{(t+1)}$ equal to $\Theta$ which maximizes $Q_\beta(\Theta; \Theta^{(t)})$.
4. Increase $\beta$.
5. If $\beta < \beta_{max}$, set $t \leftarrow t + 1$, and repeat from step 3; otherwise stop.

One can see that in the proposed algorithm, an outer loop is added to the original EM algorithm for the annealing process. An important distinction to keep in mind is that unlike simulated annealing, the optimization in step 3 is *deterministically* performed at each $\beta$. Now let's consider the effect of the posterior parameterization of Eq. 10. The annealing process begins at small $\beta$ (high temperature). Clearly, at this time, since $f(y_k|x_k)$ becomes uniform, $-F_\beta(\Theta)$ has only one global maximum. Hence, the maximum can be easily found. Then by gradually increasing $\beta$, the influence of each $x_k$ is gradually localized. At $\beta > 0$, function $Q_\beta$ will have several local maxima. However, at each step, it can be assumed that the new global maximum is close to the previous one. Hence, by this assumption, the algorithm can track the global maximum at each $\beta$ while increasing $\beta$. Clearly, when $\beta = 1$ the parameterized posterior coincides with the original one. Moreover, noting that $-F_1(\Theta) \equiv L(\Theta)$, $\beta_{max}$ ought be one.

## 4 Demonstration

To visualize how the proposed algorithm works, we consider a simple one-dimensional, two-component normal mixture problem. The mixture is given by $p(x; m_1, m_2) = 0.3\frac{1}{\sqrt{2\pi}} \exp\{-\frac{1}{2}(x - m_1)^2\} + 0.7\frac{1}{\sqrt{2\pi}} \exp\{-\frac{1}{2}(x - m_2)^2\}$. In this case, $\Theta = (m_1, m_2)$, $y_k \in \{1, 2\}$, and therefore, the joint density $p(x, 1; m_1, m_2) = 0.3\frac{1}{\sqrt{2\pi}} \exp\{-\frac{1}{2}(x - m_1)^2\}$, while $p(x, 2; m_1, m_2) = 0.7\frac{1}{\sqrt{2\pi}} \exp\{-\frac{1}{2}(x - m_2)^2\}$.

One hundred samples in total were generated from this mixture with $m_1 = -2$ and $m_2 = 2$. Figure 1 shows contour plots of the $-F_\beta(m_1, m_2)/N$ surface. It is interesting to see how $F_\beta(m_1, m_2)$ varies with $\beta$. One can see that a finer and truer structure emerges by increasing $\beta$. Note that as explained before, when $\beta = 1$,

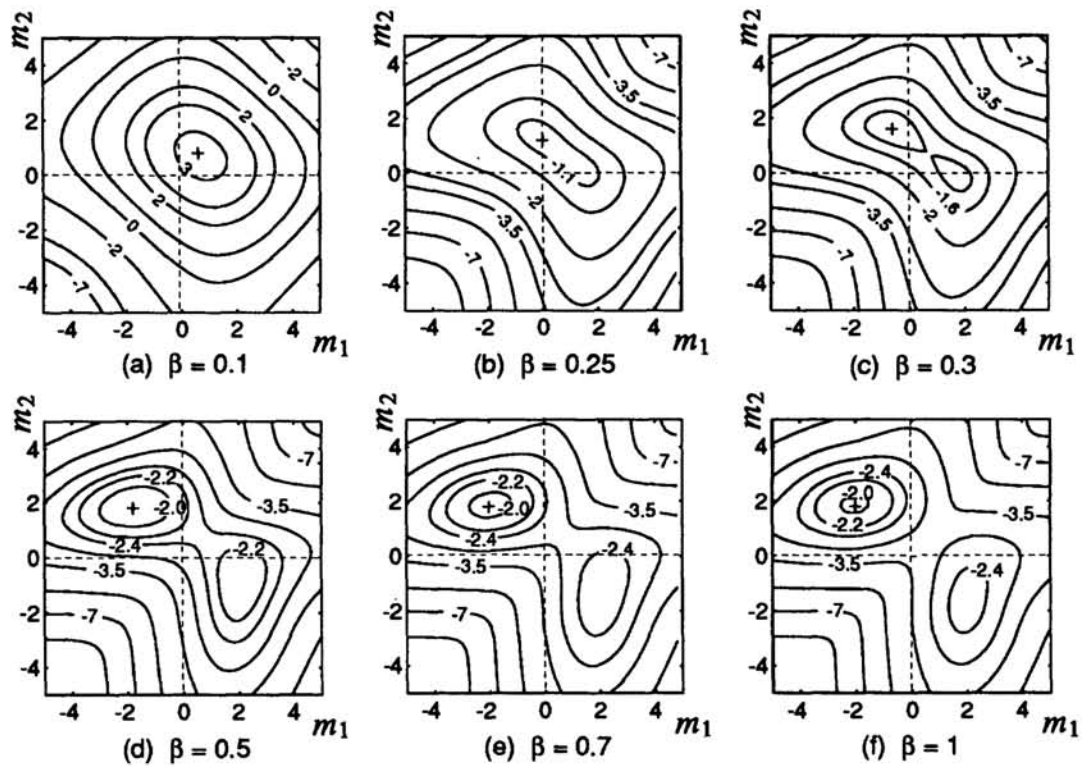

Figure 1: Contour plots of $-F_\beta(m_1, m_2)/N$ surface.
("+" denotes a global maximum at each $\beta$.)

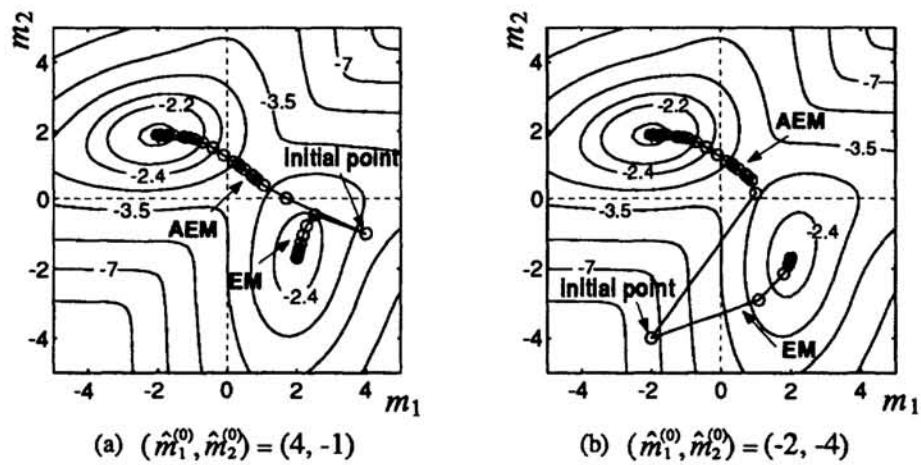

Figure 2: Trajectories for EM and AEM procedures.

$-F_1(m_1, m_2) \equiv L(m_1, m_2)$. The maximum of $-F_1$ (or $L$) occurs at $\hat{m}_1 = -2.0$ and $\hat{m}_2 = 1.9$. As initial points, $(\hat{m}_1^{(0)}, \hat{m}_2^{(0)}) = (4, -1), (-2, -4)$ were tested. Note that these points are close to a local maximum. Figure 2 shows how both algorithms converge from these starting points[3].

The original EM algorithm converges to $(\hat{m}_1^{(12)}, \hat{m}_2^{(12)}) = (2.002, -1.687)$ (Figure 2(a)) and to $(\hat{m}_1^{(19)}, \hat{m}_2^{(19)}) = (1.999, -1.696)$ (Figure 2(b)). In contrast, the proposed AEM algorithm converges to $(\hat{m}_1^{(43)}, \hat{m}_2^{(43)}) = (-2.022, 1.879)$ (Figure 2(a)) and $(\hat{m}_1^{(43)}, \hat{m}_2^{(43)}) = (-2.022, 1.879)$ (Figure 2(b)). Since these starting points are close to the local maximum point, the original EM algorithm becomes trapped by the local maximum point, while the proposed algorithm successfully converges to the global maximum in both cases.

## 5  Discussion

A new view of the EM algorithm has recently been presented (Hathaway, 1986; Neal & Hinton, 1993). This view states that the EM steps can be regarded as a grouped version of the method of *coordinate ascent* of the following objective function:

$$J \stackrel{\text{def}}{=} \sum_k^N E_{p(y_k)}\{\log p(x_k, y_k; \Theta)\} - \sum_k^N E_{p(y_k)}\{\log p(y_k)\} \qquad (15)$$

over $p(y_k)$ and $\Theta$. That is, the E-step corresponds to the maximization of $J$ with respect to $p(y_k)$ with fixed $\Theta$, while the M-step corresponds to that of $J$ with respect to $\Theta$ with fixed $p(y_k)$. Neal & Hinton mentioned that apart from the sign change, $J$ is analogous to the "free energy" well known in statistical physics.

It is worth mentioning another interpretation of the derived parameterized posterior; i.e., it plays a central role in the AEM algorithm. Taking the logarithm on both sides of Eq. 10, we have

$$-\frac{1}{\beta}\log\int p(x_k, y_k; \Theta)^\beta dy_k = -\log p(x_k, y_k; \Theta) + \frac{1}{\beta}\log f(y_k|x_k). \qquad (16)$$

Moreover, taking the conditional expectation[4] given $x_k$, summing over $k$, and using Eq. 12, we have

$$F_\beta(\Theta) = \sum_{k=1}^N E_{f(y_k|x_k)}\{-\log p(x_k, y_k; \Theta)\} + \frac{1}{\beta}\sum_{k=1}^N E_{f(y_k|x_k)}\{\log f(y_k|x_k)\}. \qquad (17)$$

From Eqs. 7 and 8, Eq. 17 can be rewritten as $F_\beta(\Theta) = E - \frac{1}{\beta}H$. Noting that $1/\beta$ corresponds to the "temperature", one can see that this expression exactly agrees with the *free energy*, while Eq. 15 is completely without the "temperature". In other words, $F_\beta(\Theta)$ can be interpreted as an annealing variant of $-J$. Clearly, when $\beta = 1$, $F_\beta(\Theta) \equiv -J$.

The proposed algorithm is also applicable to the learning of the (Generalized) Radial Basis Function (RBF, GRBF) networks. Indeed, Nowlan (Nowlan, 1990), for instance, proposes a maximum likelihood competitive learning algorithm for the RBF networks. In his study, "*soft competition*" and "*hard competition*" are experimentally compared and it is shown that the soft competition can give better performance. In our algorithm, on the other hand, the soft model exactly corresponds to the case $\beta = 1$, while the hard model corresponds to the case $\beta \rightarrow \infty$. Consequently, both models can be regarded as special cases in our algorithm.

## Acknowledgements

We would like to thank Dr. Tsukasa Kawaoka, NTT CS labs, for his encouragement.

## Footnotes

[1] In such unsupervised learning as mixture problems, $y$ reduces to an integer value ($y \in \{1, 2, \ldots, C\}$, where $C$ is the number of components), indicating the component from which a data sample $x$ originates.

[2]When the sequence converges to a saddle point (e.g., when the Hessian matrix of $-F_\beta(\Theta)$ has at least one positive eigen value), a local line search in the direction of the eigen vector corresponding to the largest eigen value should be performed to escape the solution from the saddle point.

[3]Although the AEM procedure was actually performed for successive $\beta$ ($\beta_{new} \leftarrow \beta \times 1.4$), the results are superimposed on $-F_1(m_1, m_2)/N$ surface for convenience.

[4]Since the LHS of Eq. 16 is independent of $y_k$, it does not change after expectation.

## References

J. Buhmann & H. Kuhnel. (1993) Complexity optimized data clustering by competitive neural networks. *Neural Computation*, 5:75–88.

W. Byrne. (1992) Alternating minimization and Boltzmann machine learning. *IEEE Trans. Neural Networks*, 3:612-620.

A. P. Dempster, N. M. Laird & D. B. Rubin. (1977) Maximum-likelihood from incomplete data via the EM algorithm. *J. Royal Statist. Soc. Ser. B (methodological)*, 39:1-38.

S. Geman & D. Geman. (1984) Stochastic relaxation, Gibbs distribution and the Baysian restortion in images. *IEEE Trans. Pattern Anal. Machine Intell.*, 6,6:721-741.

R. J. Hathaway. (1986) Another interpretation of the EM algorithm for mixture distributions. *Statistics & Probability Letters*, 4: 53-56.

M. I. Jordan & R. A. Jacob. (1993) Hierarchical mixtures of experts and the EM algorithm. MIT Dept. of Brain and Cognitive Science preprint.

R. M. Neal & G. E. Hinton. (1993) A new view of the EM algorithm that justifies incremental and other variants. submitted to *Biometrika*.

S. J. Nowlan. (1990) Maximum likelihood competitve learning. in D. S. Touretzky *et al.* eds., *Advances in Neural Information Systems* 2, Morgan Kaufmann. 574-582.

K. Rose, E. Gurewitz & G. C. Fox. (1992) Vector quantization by deterministic annealing. *IEEE Trans. Information Theory*, 38,4:1249-1257.

N. Ueda & R. Nakano. (1994) Mixture density estimation via EM algorithm with deterministic annealing. in proc. *IEEE Neural Networks for Signal Processing*, 69-77.

Y. Wong. (1993) Clustering data by melting. *Neural Computation*, 5:89-104.

A. L. Yuille, P. Stolorz & J. Utans. (1994) Statistical physics, mixtures of distributions, and the EM algorithm. *Neural Computation*, 6:334-340.
